# Collaborative Ranking With 17 Parameters

**Maksims N. Volkovs**
University of Toronto
mvolkovs@cs.toronto.edu

**Richard S. Zemel**
University of Toronto
zemel@cs.toronto.edu

## Abstract

The primary application of collaborate filtering (CF) is to recommend a small set of items to a user, which entails ranking. Most approaches, however, formulate the CF problem as rating prediction, overlooking the ranking perspective. In this work we present a method for collaborative ranking that leverages the strengths of the two main CF approaches, neighborhood- and model-based. Our novel method is highly efficient, with only seventeen parameters to optimize and a single hyperparameter to tune, and beats the state-of-the-art collaborative ranking methods. We also show that parameters learned on datasets from one item domain yield excellent results on a dataset from very different item domain, without any retraining.

## 1  Introduction

Collaborative Filtering (CF) is a method of making predictions about an individual's preferences based on the preference information from many users. The emerging popularity of web-based services such as Amazon, YouTube, and Netflix has led to significant developments in CF in recent years. Most applications use CF to recommend a small set of items to the user. For instance, Amazon presents a list of top-T products it predicts a user is most likely to buy next. Similarly, Netflix recommends top-T movies it predicts a user will like based on his/her rating and viewing history.

However, while recommending a small ordered list of items is a ranking problem, ranking in CF has gained relatively little attention from the learning-to-rank community. One possible reason for this is the Netflix[3] challenge which was the primary venue for CF model development and evaluation in recent years. The challenge was formulated as a rating prediction problem, and almost all of the proposed models were designed specifically for this task, and were evaluated using the normalized squared error objective. Another potential reason is the absence of user-item features. The standard learning-to-rank problem in information retrieval (IR), which is well explored with many powerful approaches available, always includes item features, which are used to learn the models. These features incorporate a lot of external information and are highly engineered to accurately describe the query-document pairs. While a similar approach can be taken in CF settings, it is likely to be very time consuming to develop analogous features, and features developed for one item domain (books, movies, songs etc.) are likely to not generalize well to another. Moreover, user features typically include personal information which cannot be publicly released, preventing open research in the area. An example of this is the second part of the Netflix challenge which had to be shut down due to privacy concerns. The absence of user-item features makes it very challenging to apply the models from the learning-to-rank domain to this task. However, recent work [23, 15, 2] has shown that by optimizing a ranking objective just given the known ratings a significantly higher ranking accuracy can be achieved as compared to models that optimize rating prediction.

Inspired by these results we propose a new ranking framework where we show how the observed ratings can be used to extract effective feature descriptors for every user-item pair. The features do not require any external information and make it it possible to apply any learning-to-rank method to optimize the parameters of the ranking function for the target metric. Experiments on MovieLens and Yahoo! datasets show that our model outperforms existing rating and ranking approaches to CF.

Moreover, we show that a model learned with our approach on a dataset from one user/item domain can then be applied to a different domain *without* retraining and still achieve excellent performance.

## 2 Collaborative Ranking Framework

In a typical collaborative filtering (CF) problem we are given a set of $N$ users $\mathbf{U} = \{u_1, ..., u_N\}$ and a set of $M$ items $\mathbf{V} = \{v_1, ..., v_M\}$. The users' ratings of the items can be represented by an $N \times M$ matrix $\mathbf{R}$ where $\mathbf{R}(u_n, v_m)$ is the rating assigned by user $u_n$ to item $v_m$ and $\mathbf{R}(u_n, v_m) = 0$ if $v_m$ is not rated by $u_n$. We use $\mathbf{U}(v_m)$ to denote the set of all users that have rated $v_m$ and $\mathbf{V}(u_n)$ to denote the set of items that have been rated by $u_n$. We use vector notation: $\mathbf{R}(u_n, :)$ denotes the $n$'th row of $\mathbf{R}$ ($1 \times M$ vector), and $\mathbf{R}(:, v_m)$ denotes the $m$'th column ($N \times 1$ vector).

As mentioned above, most research has concentrated on the rating prediction problem in CF where the aim is to accurately predict the ratings for the unrated items for each user. However, most applications that use CF typically aim to recommend only a small ranked set of items to each user. Thus rather than concentrating on rating prediction we instead approach this problem from the ranking viewpoint and refer to it as Collaborative Ranking (CR). In CR the goal is to rank the unrated items in the order of relevance to the user. A ranking of the items $\mathbf{V}$ can be represented as a permutation $\pi : \{1, ..., M\} \rightarrow \{1, ..., M\}$ where $\pi(m) = l$ denotes the rank of the item $v_m$ and $m = \pi^{-1}(l)$. A number of evaluation metrics have been proposed in IR to evaluate the performance of the ranking. Here we use the most commonly used metric, Normalized Discounted Cumulative Gain (NDCG) [12]. For a given user $u_n$ and ranking $\pi$ the NDCG is given by:

$$NDCG(u_n, \pi, \mathbf{R})@T = \frac{1}{G_T(u_n, \mathbf{R})} \sum_{t=1}^{T} \frac{2^{\mathbf{R}(u_n, v_{\pi^{-1}(t)})} - 1}{\log(t+1)} \tag{1}$$

where $T$ is a truncation constant, $v_{\pi^{-1}(t)}$ is the item in position $t$ in $\pi$ and $G_T(u_n, \mathbf{R})$ is a normalizing term which ensures that $NDCG \in [0, 1]$ for all rankings. $T$ is typically set to a small value to emphasize that the user will only be shown the top-T ranked items and the items below the top-T are not evaluated.

## 3 Related Work

Related work in CF and CR can be divided into two categories: neighborhood-based approaches and model-based approaches. In this section we describe both types of models.

### 3.1 Neighborhood-Based Approaches

Neighborhood-based CF approaches estimate the unknown ratings for a target user based on the ratings from the set of neighborhood users that tend to rate similarly to the target user. Formally, given the target user $u_n$ and item $v_m$ the neighborhood-based methods find a subset of $K$ neighbor users who are most similar to $u_n$ and have rated $v_m$, i.e., are in the set $\mathbf{U}(v_m) \setminus u_n$. We use $\mathbf{K}(u_n, v_m) \subseteq \mathbf{U}(v_m) \setminus u_n$ to denote the set of $K$ neighboring users. A central component of these methods is the similarity function $\psi$ used to compute the neighbors. Several such functions have been proposed including the Cosine Similarity [4] and the Pearson Correlation [20, 10]:

$$\psi_{cos}(u_n, u') = \frac{\mathbf{R}(u_n, :) \cdot \mathbf{R}(u', :)^T}{\|\mathbf{R}(u_n, :)\| \|\mathbf{R}(u', :)\|} \qquad \psi_{pears}(u_n, u') = \frac{(\mathbf{R}(u_n, :) - \mu(u_n)) \cdot (\mathbf{R}(u', :) - \mu(u'))^T}{\|\mathbf{R}(i, :) - \mu(u_n)\| \|\mathbf{R}(u', :) - \mu(u')\|}$$

where $\mu(u_n)$ is the average rating for $u_n$. Once the $K$ neighbors are found the rating is predicted by taking the weighted average of the neighbors' ratings. An analogous item-based approach [22] can be used when the number of items is smaller than the number of users.

One problem with the neighborhood-based approaches is that the raw ratings often contain user bias. For instance, some users tend to give high ratings while others tend to give low ones. To correct for these biases various methods have been proposed to normalize or center the ratings [4, 20] before computing the predictions.

Another major problem with the neighborhood-based approaches arises from the fact that the observed rating matrix $\mathbf{R}$ is typically highly sparse, making it very difficult to find similar neighbors reliably. To addresss this sparsity, most methods employ dimensionality reduction [9] and data smoothing [24] to fill in some of the unknown ratings, or to cluster users before computing user

similarity. This however adds computational overhead and typically requires tuning additional parameters such as the number of clusters.

A neighborhood-based approach to ranking has been proposed recently by Liu & Yang [15]. Instead of predicting ratings, this method uses the neighbors of $u_n$ to fill in the missing entries in the $M \times M$ pairwise preference matrix $\mathbf{Y}_n$, where $\mathbf{Y}_n(v_m, v_l)$ is the preference strength for $v_m$ over $v_l$ by $u_n$. Once the matrix is completed an approximate Markov chain algorithm is used to infer the ranking from the pairwise preferences. The main drawback of this approach is that the model is not optimized for the target evaluation metric, such as NDCG. The ranking is inferred directly from $\mathbf{Y}_n$ and no additional parameters are learned. In general, to the best of our knowledge, no existing neighborhood-based CR method takes the target metric into account during optimization.

### 3.2 Model-Based Approaches

In contrast to the neighborhood-based approaches, the model-based approaches use the observed ratings to create a compact model of the data which is then used to predict the unobserved ratings. Methods in this category include latent models [11, 16, 21], clustering methods [24] and Bayesian networks [19]. Latent factorization models such as Probabilistic Matrix Factorization (PMF) [21] are the most popular model-based approaches. In PMF every user $u_n$ and item $v_m$ are represented by latent vectors $\phi(u_n)$ and $\phi(v_m)$ of length $D$. For a given user-item pair $(u_n, v_m)$ the dot product of the corresponding latent vectors gives the rating prediction: $\mathbf{R}(u_n, v_m) \approx \phi(u_n) \cdot \phi(v_m)$. The latent representations are learned by minimizing the squared error between the observed ratings and the predicted ones.

Latent models have more expressive power and typically perform better than the neighborhood-based models when the number of observed ratings is small because they are able to learn preference correlations that extend beyond the simple neighborhood similarity. However, this comes at the cost of a large number of parameters and complex optimization. For example, with the suggested setting of $D = 20$ the PMF model on the full Netflix dataset has over 10 million parameters and is prone to overfitting. To prevent overfitting the weighted $\ell^2$ norms of the latent representations are minimized together with the squared error during the optimization phase, which introduces additional hyper-parameters to tune.

Another problem with the majority of the model-based approaches is that inference for a new user/item is typically expensive. For instance, in PMF the latent representation has to be learned before any predictions can be made for a new user/item, and if many new users/items are added the entire model has to be retrained. On the other hand, inference for a new user in neighborhood-based methods can be done efficiently by simply computing the $K$ neighbors, which is a key advantage of these approaches.

Several model-based approaches to CR have recently been proposed, notably CofiRank [23] and the PMF-based ranking model [2]. CofiRank learns latent representations that minimize a ranking-based loss instead of the squared error. The PMF-based approach uses the latent representations produced by PMF as user-item features and learns a ranking model on these features. The authors of that work also note that the PMF representations might not be optimal for ranking since they are learned using a squared error objective which is very different from most ranking metric. To account for this they propose an extension where both user-item features and the weights of the ranking function are optimized during learning. Both methods incorporate NDCG during the optimization phase which is a significant advantage over most neighborhood-based approaches to CR. However, neither method addresses the optimization or inference problems mentioned above. In the following section we present our approach to CR which leverages the advantages of both neighborhood and model-based methods.

### 3.3 Learning-to-Rank

Learning-to-rank has received a lot of attention in the machine learning community due to its importance in a wide variety of applications ranging from information retrieval to natural language processing to computer vision. In IR the learning-to-rank problem consists of a set of training queries where for each query we are given a set of retrieved documents and their relevance labels that indicate the degree of relevance to the query. The documents are represented as query dependent feature vectors and the goal is to learn a feature-based ranking function to rank the documents in the order of relevance to the query. Existing approaches to this problem can be partitioned into three

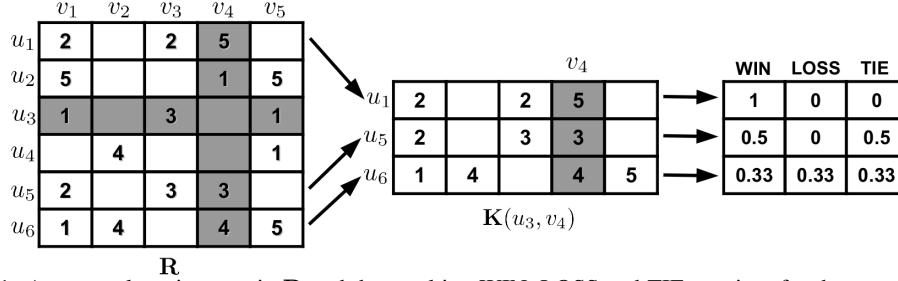

Figure 1: An example rating matrix $\mathbf{R}$ and the resulting WIN, LOSS and TIE matrices for the user-item pair $(u_3, v_4)$ with $K = 3$ (number of neighbors). (1) Top-3 closest neighbors $\{u_1, u_5, u_6\}$ are selected from $\mathbf{U}(v_4) = \{u_1, u_2, u_5, u_6\}$ (all users who rated $v_4$). Note that $u_2$ is not selected because the ratings for $u_2$ deviate significantly from those for $u_3$. (2) The WIN, LOSS and TIE matrices are computed for each neighbor using Equation 2. Here $g \equiv 1$ is used to compute the matrices. For example, $u_5$ gave a rating of 3 to $v_4$ which ties it with $v_3$ and beats $v_1$. Normalizing by $|\mathbf{V}(u_5)| - 1 = 2$ gives $\text{WIN}_{34}(u_5) = 0.5$, $\text{LOSS}_{34}(u_5) = 0$ and $\text{TIE}_{34}(u_5) = 0.5$.

categories: pointwise, pairwise, and listwise. Due to the lack of space we omit the description of the individual approaches here and instead refer the reader to [14] for an excellent overview.

## 4 Our Approach

The main idea behind our approach is to transform the CR problem into a learning-to-rank one and then utilize one of the many developed ranking methods to learn the ranking function. CR can be placed into the learning-to-rank framework by noting that the users correspond to queries and items to documents. For each user the observed ratings indicate the relevance of the corresponding items to that user and can be used to train the ranking function. The key difference between this setup and the standard learning-to-rank one is the absence of user-item features. In this work we bridge this gap and develop a robust feature extraction approach which does not require any external user or item information and is based only on the available training ratings.

### 4.1 Feature Extraction

The PMF-based ranking approach [2] extracts user-item features by concatenating together the latent representations learned by the PMF model. The model thus requires the user-item representations to be learned before the items can be ranked and hence suffers from the main disadvantages of the model-based approaches: the large number of parameters, complex optimization, and expensive inference for new users and items. In this work we take a different approach which avoids these disadvantages. We propose to use the neighbor preferences to extract the features for a given user-item pair.

Formally, given a user-item pair $(u_n, v_m)$ and a similarity function $\psi$, we use $\psi$ to extract a subset of the $K$ most similar users to $u_n$ that rated $v_m$, i.e., $\mathbf{K}(u_n, v_m)$. This step is identical to the standard neighborhood-based model, and $\psi$ can be any rating or preference based similarity function. Once $\mathbf{K}(u_n, v_m) = \{u_k\}_{k=1}^{K}$ is found, instead of using only the ratings for $v_m$, we use *all* of the observed ratings for each neighbor and summarize the net preference for $v_m$ into three $K \times 1$ summary preference matrices $\text{WIN}_{nm}$, $\text{LOSS}_{nm}$ and $\text{TIE}_{nm}$:

$$\text{WIN}_{nm}(k) = \frac{1}{|\mathbf{V}(u_k)| - 1} \sum_{v' \in \mathbf{V}(u_k) \setminus v_m} g(\mathbf{R}(u_k, v_m), \mathbf{R}(u_k, v'))I[\mathbf{R}(u_k, v_m) > \mathbf{R}(u_k, v')]$$

$$\text{LOSS}_{nm}(k) = \frac{1}{|\mathbf{V}(u_k)| - 1} \sum_{v' \in \mathbf{V}(u_k) \setminus v_m} g(\mathbf{R}(u_k, v_m), \mathbf{R}(u_k, v'))I[\mathbf{R}(u_k, v_m) < \mathbf{R}(u_k, v')] \quad (2)$$

$$\text{TIE}_{nm}(k) = \frac{1}{|\mathbf{V}(u_k)| - 1} \sum_{v' \in \mathbf{V}(u_k) \setminus v_m} I[\mathbf{R}(u_k, v_m) = \mathbf{R}(u_k, v')]$$

where $I[x]$ is an indicator function evaluating to 1 if $x$ is true and to 0 otherwise, and $g : \mathbb{R}^2 \to \mathbb{R}$ is the pairwise preference function used to convert ratings to pairwise preferences. A simple choice for $g$ is $g \equiv 1$ which ignores the rating magnitude and turns the matrices into normalized counts. However, recent work in preference aggregation [8, 13] has shown that additional gain can be achieved by taking the relative rating magnitude into account by using either the normalized rating or log rating difference. All three versions of $g$ address the user bias problem mentioned above by using

relative comparisons rather than the absolute rating magnitude. In this form $\text{WIN}_{nm}(k)$ corresponds to the net positive preference for $v_m$ by neighbor $u_k$. Similarly, $\text{LOSS}_{nm}(k)$ corresponds to the net negative preference and $\text{TIE}_{nm}(k)$ counts the number of ties. Together the three matrices thus describe the relative preferences for $v_m$ across all the neighbors of $u_n$. Normalization by $|\mathbf{V}(u_k) \setminus v_m|$ (number of observed ratings for $u_k$ excluding $v_m$), ensures that the entries are comparable across neighbors with different numbers of ratings. For unpopular items $v_m$ that do not have many ratings with $|\mathbf{U}(v_m)| < K$, the number of neighbors will be less than $K$, i.e., $|\mathbf{K}(u_n, v_m)| < K$. When such an item is encountered we shrink the preference matrices to be the same size as $|\mathbf{K}(u_n, v_m)|$. Figure 1 shows an example rating matrix $\mathbf{R}$ together with the preference matrices computed for the user-item pair $(u_3, v_4)$.

Given the preference matrix $\text{WIN}_{nm}$ we summarize it with a set of simple descriptive statistics:

$$\gamma(\text{WIN}_{nm}) = \left[ \mu(\text{WIN}_{nm}), \ \sigma(\text{WIN}_{nm}), \ \max(\text{WIN}_{nm}), \ \min(\text{WIN}_{nm}), \ \frac{1}{K}\sum_k I[\text{WIN}_{nm}(k) \neq 0] \right]$$

where $\mu$ and $\sigma$ are mean and standard deviation functions respectively. The last statistic counts the number of neighbors (out of $K$) that express any positive preference towards $v_m$, and together with $\sigma$ summarizes the overall confidence of the preference. Extending this procedure to the other two preference matrices and concatenating the resulting statistics gives the feature vector for $(u_n, v_m)$:

$$\gamma(u_n, v_m) = [\gamma(\text{WIN}_{nm}), \gamma(\text{LOSS}_{nm}), \gamma(\text{TIE}_{nm})] \tag{3}$$

Intuitively the features describe the net preference for $v_m$ and its variability across the neighbors. Note that since $\gamma$ is independent of $K$, $N$ and $M$ this representation will have the *same* length for every user-item pair. We have thus created a fixed length *feature representation* for every user-item pair, effectively transforming the CR problem into a standard learning-to-rank one. During training our aim is now to use the observed training ratings to learn a scoring function $f : \mathbb{R}^{|\gamma|} \to \mathbb{R}$ which maximizes the target IR metric, such as NDCG, across all users. At test time, given a user $u$ and items $\{v_1, ..., v_M\}$, we (1) extract features for each item $v_m$ using the neighbors of $(u, v_m)$; (2) apply the learned scoring function to get the score for every item; and (3) sort the scores to produce the ranking. This process is shown in Figure 2.

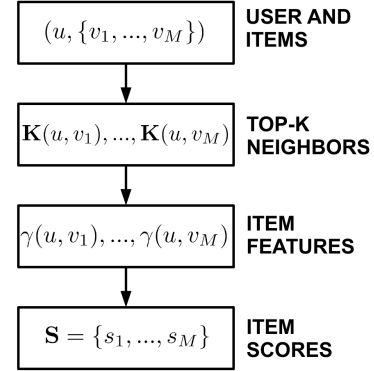

USER AND ITEMS

$(u, \{v_1, ..., v_M\})$

TOP-K NEIGHBORS

$\mathbf{K}(u, v_1), ..., \mathbf{K}(u, v_M)$

ITEM FEATURES

$\gamma(u, v_1), ..., \gamma(u, v_M)$

ITEM SCORES

$\mathbf{S} = \{s_1, ..., s_M\}$

Figure 2: The flow diagram for WLT, our feature-based CR model.

It is important to note here that, first, a *single* scoring function is learned for all users and items so the number of parameters is independent of the number of users or items and only depends on the size of $\gamma$. This is a significant advantage over most model-based approaches where the number of parameters typically scales linearly with the number of users and/or items. Second, given a new user $u$ no optimization is necessary to produce a ranking of the items for $u$. Similarly to neighborhood-based methods, our approach only requires computing the neighbors to extract the features and apply the learned scoring function to get the ranking. This is also a significant advantage over most user-based approaches where it is typically necessary to learn a new model for every user not present in the training data before predictions can be made. Finally, unlike the existing neighborhood-based methods to CR our approach allows to optimize the parameters of the model for the target metric. Moreover, the extracted features incorporate preference confidence information such as the variance across the neighbors and the fraction of the neighbors that generated each preference type (positive, negative and tie). Taking this information into account allows us to adapt the parameters of the scoring function to sparse low-confidence settings and addresses the reliability problem of the neighborhood-based methods (see Section 3.1). Note that an analogous item-based approach can be taken here by similarly summarizing the preferences of $u_n$ for items that are closest to $v_m$, we leave this for future work. A modified version of this approach adapted to binary ratings recently placed second in the Million Song Dataset Challenge [18] ran by Kaggle.

## 4.2 Learning the Scoring Function

Given the user-item features extracted based on the neighbors our goal is to use the observed training ratings for each user to optimize the parameters of the scoring function for the target IR metric. A key difference between this feature-based CR approach and the typical learning-to-rank setup is the

possibility of missing features. If a given training item $v_m$ is not ranked by any other user except $u_n$ the feature vector is set to zero ($\gamma(u_n, v_m) \equiv 0$). One way to avoid missing features is to learn only with those items that have at least $\epsilon$ ratings in the training set. However, in very sparse settings this would force us to discard some of the valuable training data. We take a different approach, modifying the conventional linear scoring function to include an additional bias term $b_0$:

$$f(\gamma(u_n, v_m), \mathbf{W}) = \mathbf{w} \cdot \gamma(u_n, v_m) + b + I[\mathbf{U}(v_m) \setminus u_n = \emptyset] b_0 \qquad (4)$$

where $\mathbf{W} = \{\mathbf{w}, b, b_0\}$ is the set of free parameters to be learned. Here $w$ has the same dimension as $\gamma$, and $I$ is an indicator function. The bias term $b_0$ provides a base score for $v_m$ if $v_m$ does not have enough ratings in the training data. Several possible extensions of this model are worth mentioning here. First, the scoring function can be made non-linear by adding additional hidden layer(s) as done in conventional multilayer neural networks. Second, user information can be incorporated into the model by learning user specific weights. To incorporate user information we can learn a separate set of weights $w_n$ for each user $u_n$ or group of users. The weights will provide user specific information and are then applied to rank the unrated items for the corresponding user(s). However, this extension makes the approach similar to the model-based approaches, with all the corresponding disadvantages mentioned above. Finally, additional user/item information such as, for example, personal information for users and description/genre etc. for items, can be incorporated by simply concatenating it with $\gamma(u_n, v_m)$ and expanding the dimensionality of W. Note that if these additional features can be extracted efficiently, incorporating them will not add significant overhead to either learning or inference and the model can still be applied to new users and items very efficiently.

In the form given by Equation 4 our model has a total of $|\gamma| + 2$ parameters to be learned. We can use any of the developed learning-to-rank approaches to optimize W. In this work we chose to use the LambdaRank method, due it its excellent performance, having recently won the Yahoo! Learning-To-Rank Challenge [7]. We omit the description of LambdaRank here due to the lack of space, and refer the reader to [6] and [5] for a detailed description.

## 5   Experiments

To validate the proposed approach we conducted extensive experiments on three publicly available datasets: two movie datasets MovieLens-1, MovieLens-2, and a musical artist dataset from Yahoo! [1]. All datasets were kept as is except Yahoo!, which we subsampled by first selecting the 10,000 most popular items and then selecting the 100,000 users with the most ratings. The subsampling was done to speed up the experiments as the original dataset has close to 2 million users and 100,000 items. In addition to subsampling we rescaled user ratings from 0-100 to the 1-5 interval to make the data consistent with the other two datasets. The rescaling was done by mapping 0-19 to 1, 20-39 to 2, etc. The user, item and rating statistics are summarized in Table 1. To investigate the effect that the number of ratings has on accuracy we follow the framework of [23, 2].

For each dataset we randomly select 10, 20, 30, 40 ratings from each user for training, 10 for validation and test on the remaining ratings. Users with less than 30, 40, 50, 60 ratings were removed to ensure that we could evaluate on at least 10 ratings for each user. Note that the number of test items varies significantly across users with many

Table 1: Dataset statistics.

| Dataset | Users | Items | Ratings |
|---|---|---|---|
| MovieLens-1 | 1000 | 1700 | 100,000 |
| MovieLens-2 | 72,000 | 10,000 | 10,000,000 |
| Yahoo! | 100,000 | 10,000 | 45,729,723 |

users having more test ratings than training ones. This simulates the real life CR scenario where the set of unrated items from which the recommendations are generated is typically much larger than the rated item set for each user.

We trained our ranking model, referred to as WLT, using stochastic gradient descent with the learning rates $10^{-2}$, $10^{-3}$, $10^{-4}$ for MovieLens-1, MovieLens-2 and Yahoo! respectively. We found that 1 to $2^1$ iterations was sufficient to trained the models. We also found that using smaller learning rates typically resulted in better generalization. We compare WLT with a well established user-based (UB) collaborative filtering model. We also compare with two collaborative ranking models: PMF-based ranker [2] (PMF-R) and CofiRank [23] (CO). To make the comparison fair we used the same LambdaRank architecture to train both WLT and PMF-R. Note that both PMF-R and CofiRank report state-of-the-art CR results. To compute the PMF features we used extensive cross-validation to determine the $L^2$ penalty weights and the latent dimension size $D$ (5, 10, 10 for MovieLens-1, MovieLens-2, and Yahoo! datasets respectively). For CofiRank we used the settings suggested

Table 2: Collaborative Ranking results. NDCG values at different truncation levels are shown within the main columns, which are split based on the number of training ratings. Each model's rounded number of parameters is shown in brackets, with K = thousand, M = million.

| | 10 | | | 20 | | | 30 | | | 40 | | |
|---|---|---|---|---|---|---|---|---|---|---|---|---|
| | N@1 | N@3 | N@5 | N@1 | N@3 | N@5 | N@1 | N@3 | N@5 | N@1 | N@3 | N@5 |
| **MovieLens-1**: | | | | | | | | | | | | |
| UB | 49.30 | 54.67 | 57.36 | 57.49 | 61.81 | 62.88 | 64.25 | 65.75 | 66.58 | 62.27 | 64.92 | 66.14 |
| PMF-R(12K) | 69.39 | 68.33 | **68.65** | **72.50** | 70.42 | 69.95 | **72.77** | **72.23** | **71.55** | 74.02 | 71.55 | 70.90 |
| CO(240K) | 67.28 | 66.23 | 66.59 | 71.82 | **70.80** | **70.30** | 71.60 | 71.15 | 70.58 | 71.43 | 71.64 | 71.43 |
| WLT(17) | **70.96** | **68.25** | 67.98 | 70.34 | 69.50 | 69.21 | 71.41 | 71.16 | 71.02 | **74.09** | **71.85** | **71.52** |
| **MovieLens-2**: | | | | | | | | | | | | |
| UB | 67.62 | 68.23 | 68.74 | 71.29 | 70.78 | 70.87 | 72.65 | 71.98 | 71.90 | 73.33 | 72.63 | 72.42 |
| PMF-R(500K) | 70.12 | 69.41 | 69.35 | 70.65 | 70.04 | 70.09 | 72.22 | 71.48 | 71.43 | 72.18 | 71.60 | 71.55 |
| CO(5M) | 70.14 | 68.40 | 68.46 | 68.80 | 68.51 | 68.76 | 64.60 | 65.62 | 66.38 | 62.82 | 63.49 | 64.25 |
| WLT(17) | **72.78** | **71.70** | **71.49** | **73.93** | **72.63** | **72.37** | **74.67** | **73.37** | **73.04** | **75.19** | **73.73** | **73.30** |
| **Yahoo!**: | | | | | | | | | | | | |
| UB | 57.20 | 55.29 | 54.31 | 64.29 | 61.48 | 60.16 | 66.82 | 63.83 | 62.42 | 68.97 | 65.89 | 64.50 |
| PMF-R(1M) | 52.86 | 51.98 | 51.53 | 63.93 | 62.42 | **61.65** | 66.82 | 65.41 | 64.61 | 69.46 | 68.05 | 67.21 |
| CO(10M) | 57.42 | **56.88** | **56.46** | 60.59 | 59.94 | 59.48 | 62.07 | 61.10 | 60.54 | 61.68 | 60.78 | 60.24 |
| WLT(17) | **58.76** | 55.20 | 53.53 | **66.06** | **62.77** | 61.21 | **69.74** | **66.58** | **65.02** | **71.50** | **68.52** | **67.00** |

in [23] and ran the code available on the author's home page. Similarly to [2], we found that the regression-based objective almost always gave the best results for CofiRank, consistently outperforming NDCG and ordinal objectives.

For WLT and UB models we use cosine similarity as the distance function to find the top-$K$ neighbors. Note that using the same similarity function ensures that both models select the same neighbor sets and allows for fair comparison. The number of neighbors $K$ was cross validated in the range $[10, 100]$ on the small MovieLens-1 dataset and set to 200 on all other datasets as we found the results to be insensitive for $K$ above 100 which is consistent with the findings of [15]. In all experiments only ratings in the training set were used to select the neighbors, and make predictions for the validation and test set items.

## 5.1 Results

The NDCG (N@T) results at truncations 1,3 and 5 are shown in Table 2. From the table it is seen that the WLT model performs comparably to the best baseline on MovieLens-1, outperforms all methods on MovieLens-2 and is also the best overall approach on Yahoo!. Across the datasets the gains are especially large at lower truncations N@1 and N@3, which is important since those items will most likely be the ones viewed by the user.

Several patterns can also be seen from the table. First, when the number of users and ratings is small (MovieLens-1) the performance of the UB approach significantly drops. This is likely due to the fact that neighbors cannot be found reliably in this setting since users have little overlap in ratings. By taking into account the confidence information such as the number of available neighbors WLT is able to significantly improve over UB while using the *same* set of neighbors. On MovieLens-1 WLT outperforms UB by as much as 20 NDCG points. Second, for larger datasets such as MovieLens-2 and Yahoo! the model-based approaches have millions of parameters (shown in brackets in Table 2) to optimize and are highly prone to overfitting. Tuning the hyper-parameters for these models is difficult and computationally expensive in this setting as it requires conducting many cross-validation runs over large datasets. On the other hand, our approach achieves consistently better performance with only 17 parameters, and a single hyper-parameter $K$ which is fixed to 200. Overall, the results demonstrate the robustness of the proposed features which generalize well when both few and many users available.

## 5.2 Transfer Learning Results

In addition to the small number of parameters, another advantage of our approach over most model-based methods is that inference for a new user only requires finding the $K$ neighbors. Thus both users and items can be taken from a different, unseen during training, set. This transfer learning task is much more difficult than the strong generalization task [17] commonly used to test CF methods on new users. In strong generalization the models are evaluated on users not present at training time while keeping the item set fixed, while here the item set also changes. Note that it is impossible to

Table 3: Transfer learning NDCG results. Original: WLT model trained on the respective dataset. WLT-M1 and WLT-M2 models are trained on MovieLens-1 and MovieLens-2 respectively, WLT-Y is trained on Yahoo!. WLT-M1, WLT-M2 and WLT-Y models are applied to other datasets *without* retraining.

|  | 10 | | | 20 | | | 30 | | | 40 | | |
| --- | --- | --- | --- | --- | --- | --- | --- | --- | --- | --- | --- | --- |
|  | N@1 | N@3 | N@5 | N@1 | N@3 | N@5 | N@1 | N@3 | N@5 | N@1 | N@3 | N@5 |
| **MovieLens-1**: | | | | | | | | | | | | |
| Original | 70.96 | 68.25 | 67.98 | 70.34 | 69.50 | 69.21 | 71.41 | 71.16 | 71.02 | 74.09 | 71.85 | 71.52 |
| WLT-M2 | 63.15 | 62.46 | 62.75 | 69.66 | 68.61 | 68.47 | 71.02 | 70.99 | 70.88 | 73.28 | 71.70 | 71.46 |
| WLT-Y | 44.12 | 47.06 | 48.75 | 61.73 | 62.60 | 63.57 | 67.33 | 66.99 | 67.99 | 71.11 | 69.22 | 68.95 |
| **MovieLens-2**: | | | | | | | | | | | | |
| Original | 72.78 | 71.70 | 71.49 | 73.93 | 72.63 | 72.37 | 74.67 | 73.37 | 73.04 | 75.19 | 73.73 | 73.30 |
| WLT-M1 | 72.90 | 71.77 | 71.57 | 73.97 | 72.59 | 72.34 | 74.67 | 73.36 | 73.01 | 75.28 | 73.76 | 73.28 |
| WLT-Y | 68.04 | 68.03 | 68.41 | 71.54 | 71.02 | 71.07 | 73.15 | 72.38 | 72.25 | 74.00 | 73.03 | 72.79 |
| **Yahoo!**: | | | | | | | | | | | | |
| Original | 58.76 | 55.20 | 53.53 | 66.06 | 62.77 | 61.21 | 69.74 | 66.58 | 65.02 | 71.50 | 68.52 | 67.00 |
| WLT-M1 | 57.93 | 53.91 | 52.35 | 66.03 | 62.68 | 61.18 | 68.93 | 65.85 | 64.32 | 71.15 | 68.17 | 66.65 |
| WLT-M2 | 58.81 | 54.70 | 53.15 | 65.29 | 61.95 | 60.47 | 68.68 | 65.55 | 64.07 | 70.84 | 67.91 | 66.44 |

apply PMF-R, CO and most other model-based methods to this setting without re-training the entire model. Our model, on the other hand, can be applied without re-training by simply extracting the features for every new user-item pair and applying the learned scoring function to rank the items.

To test the generalization properties of the model we took the three learned WLT models (referred to as WLT-M1, WLT-M2, WLT-Y for MovieLens-1&2 and Yahoo! respectively) and applied each model to the datasets that it was not trained on. So for instance WLT-M1 was applied to MovieLens-2 and Yahoo!. Table 3 shows the transfer results for each of the datasets along with the original results for the WLT model trained on each dataset (referred to as Original). Note that none of the models were re-trained or tuned in any way. From the table it seen that our model generalizes very well to different domains. For instance, WLT-M1 trained on MovieLens-1 is able to achieve state-of-the art performance on MovieLens-2, outperforming all the baselines that were trained on MovieLens-2. Note that MovieLens-2

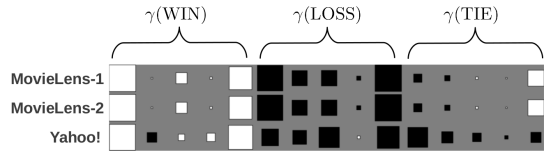

Figure 3: Normalized WLT weights. White/black correspond to positive/negative weights; the weight magnitude is proportional to the square size.

has over 5 times more items and 72 times more users than MovieLens-1, majority of which the WLT-M1 model has not seen during training. Moreover, perhaps surprisingly, our model also generalizes well across item domains. The WLT-Y model trained on musical artist data achieves state-of-the-art performance on MovieLens-2 movie data, performing better than all the baselines when 20, 30 and 40 ratings are used for training. Moreover, both WLT-M1 and WLT-M2 achieve very competitive results on Yahoo! outperforming most of the baselines.

More insight into why the model generalizes well can be gained from Figure 3, which shows the normalized weights learned by the WLT models on each of the three datsets. The weights are partitioned into feature sets from each of the three preference matrices (see Equation 2). From the figure it can be seen that the learned weights share a lot of similarities. The weights on the features from the WIN matrix are mostly positive while those on the features from the LOSS matrix are mostly negative. Mean preferences and the number of neighbors features have the highest absolute weights which indicates that they are the most useful for predicting the item scores. The similarity between the weight vectors suggests that the features convey very similar information and remain invariant across different user/item sets.

## 6   Conclusion

In this work we presented an effective approach to extract user-item features based on neighbor preferences. The features allow us to apply any learning-to-rank approach to learn the ranking function. Experimental results show that by using these features state-of-the art ranking results can be achieved. Going forward, the strong transfer results call into question whether the complex machinery developed for CF is appropriate when the true goal is recommendation, as the required information for finding the best items to recommend can be obtained from basic neighborhood statistics. We are also currently investigating additional features such as neighbors' rating overlap.

## Footnotes

[1]Note that 1 iteration of stochastic gradient descent corresponds to $|\mathbf{U}|$ weight updates.

# References

[1] The Yahoo! R1 dataset. `http://webscope.sandbox.yahoo.com/catalog.php?datatype=r`.

[2] S. Balakrishnan and S. Chopra. Collaborative ranking. In *WSDM*, 2012.

[3] J. Bennet and S. Lanning. The Netflix prize. `www.cs.uic.edu/~liub/KDD-cup-2007/NetflixPrize-description.pdf`.

[4] J. S. Breese, D. Heckerman, and C. Kadie. Empirical analysis of predictive algorithm for collaborative filtering. In *UAI*, 1998.

[5] C. J. C. Burges. From RankNet to LambdaRank to LambdaMART: An overview. Technical Report MSR-TR-2010-82, 2010.

[6] C. J. C. Burges, R. Rango, and Q. V. Le. Learning to rank with nonsmooth cost functions. In *NIPS*, 2007.

[7] O. Chapelle, Y. Chang, and T.-Y. Liu. The Yahoo! Learning To Rank Challenge. `http://learningtorankchallenge.yahoo.com`, 2010.

[8] D. F. Gleich and L.-H. Lim. Rank aggregation via nuclear norm minimization. In *SIGKDD*, 2011.

[9] K. Y. Goldberg, T. Roeder, D. Gupta, and C. Perkins. Eigentaste: A constant time collaborative filtering algorithm. *Information Retrieval*, 4(2), 2001.

[10] J. Herlocker, J. A. Konstan, and J. Riedl. An empirical analysis of design choices in neighborhood-based collaborative filtering algorithms. *Information Retrieval*, 5(4), 2002.

[11] T. Hofmann. Latent semantic models for collaborative filtering. *ACM Trans. Inf. Syst.*, 22(1), 2004.

[12] K. Jarvelin and J. Kekalainen. IR evaluation methods for retrieving highly relevant documents. In *SIGIR*, 2000.

[13] X. Jiang, L.-H. Lim, Y. Yao, and Y. Ye. Statistical ranking and combinatorial hodge theory. *Mathematical Programming*, 127, 2011.

[14] H. Li. *Learning to Rank for Information Retrieval and Natural Language Processing*. Morgan & Claypool, 2011.

[15] N. Liu and Q. Yang. Eigenrank: A ranking-oriented approach to collaborative filtering. In *SIGIR*, 2008.

[16] B. Marlin. Modeling user rating profiles for collaborative filtering. In *NIPS*, 2003.

[17] B. Marlin. Collaborative filtering: A machine learning perspective. Master's thesis, University of Toronto, 2004.

[18] B. McFee, T. Bertin-Mahieux, D. Ellis, and G. R. G. Lanckriet. The Million Song Dataset Challenge. In *WWW*, `http://www.kaggle.com/c/msdchallenge`, 2012.

[19] D. M. Pennock, E. Horvitz, S. Lawrence, and C. L. Giles. Collaborative filtering by personality diagnosis: A hybrid memory and model-based approach. In *UAI*, 2000.

[20] P. Resnick, N. Iacovou, M. Suchak, P. Bergstrom, and J. Riedl. Grouplens: An open architecture for collaborative filtering of netnews. In *CSCW*, 1994.

[21] R. Salakhutdinov and A. Mnih. Probabilistic matrix factorization. In *NIPS*, 2008.

[22] B. Sarwar, G. Karypis, J. Konstan, and J. Riedl. Item-based collaborative filtering recommendation algorithms. In *WWW*, 2001.

[23] M. Weimer, A. Karatzoglou, Q. V. Le, and A. J. Smola. CofiRank - maximum margin matrix factorization for collaborative ranking. In *NIPS*, 2007.

[24] G.-R. Xue, C. Lin, Q. Yang, W. Xi, H.-J. Zeng, Y. Yu, and Z. Chen. Scalable collaborative filtering using cluster-based smoothing. In *SIGIR*, 2005.

